# PAC-Bayes Learning of Conjunctions and Classification of Gene-Expression Data

**Mario Marchand**
IFT-GLO, Université Laval
Sainte-Foy (QC) Canada, G1K-7P4
*Mario.Marchand@ift.ulaval.ca*

**Mohak Shah**
SITE, University of Ottawa
Ottawa, Ont. Canada,K1N-6N5
*mshah@site.uottawa.ca*

## Abstract

We propose a "soft greedy" learning algorithm for building small conjunctions of simple threshold functions, called *rays*, defined on single real-valued attributes. We also propose a PAC-Bayes risk bound which is minimized for classifiers achieving a non-trivial tradeoff between sparsity (the number of rays used) and the magnitude of the separating margin of each ray. Finally, we test the soft greedy algorithm on four DNA micro-array data sets.

## 1  Introduction

An important challenge in the problem of classification of high-dimensional data is to design a learning algorithm that can often construct an accurate classifier that depends on the smallest possible number of attributes. For example, in the problem of classifying gene-expression data from DNA micro-arrays, if one can find a classifier that depends on a small number of genes and that can accurately predict if a DNA micro-array sample originates from cancer tissue or normal tissue, then there is hope that these genes, used by the classifier, may be playing a crucial role in the development of cancer and may be of relevance for future therapies.

The standard methods used for classifying high-dimensional data are often characterized as either "filters" or "wrappers". A filter is an algorithm used to "filter out" irrelevant attributes before using a base learning algorithm, such as the support vector machine (SVM), which was not designed to perform well in the presence of many irrelevant attributes. A wrapper, on the other hand, is used in conjunction with the base learning algorithm: typically removing recursively the attributes that have received a small "weight" by the classifier obtained from the base learner. The recursive feature elimination method is an example of a wrapper that was used by Guyon et al. (2002) in conjunction with the SVM for classification of micro-array data. For the same task, Furey et al. (2000) have used a filter which consists of ranking the attributes (gene expressions) as function of the difference between the positive-example mean and the negative-example mean. Both filters and wrappers have sometimes produced good empirical results but they are not theoretically justified. What we really need is a learning algorithm that has provably good guarantees in the presence of many irrelevant attributes. One of the first learning algorithms proposed by the COLT community has such a guarantee for the class of conjunc-

tions: if there exists a conjunction, that depends on $r$ out of the $n$ input attributes and that correctly classifies a training set of $m$ examples, then the greedy covering algorithm of Haussler (1988) will find a conjunction of at most $r \ln m$ attributes that makes no training errors. Note the absence of dependence on the number $n$ of input attributes. In contrast, the mistake-bound of the Winnow algorithm (Littlestone, 1988) has a logarithmic dependence on $n$ and will build a classifier on all the $n$ attributes.

Motivated by this theoretical result and by the fact that simple conjunctions of gene expression levels seems an interesting learning bias for the classification of DNA micro-arrays, we propose a "soft greedy" learning algorithm for building small conjunctions of simple threshold functions, called *rays*, defined on single real-valued attributes. We also propose a PAC-Bayes risk bound which is minimized for classifiers achieving a non-trivial tradeoff between sparsity (the number of rays used) and the magnitude of the separating margin of each ray. Finally, we test the proposed soft greedy algorithm on four DNA micro-array data sets.

## 2 Definitions

The input space $\mathcal{X}$ consists of all $n$-dimensional vectors $\mathbf{x} = (x_1, \ldots, x_n)$ where each real-valued component $x_i \in [A_i, B_i]$ for $i = 1, \ldots n$. Hence, $A_i$ and $B_i$ are, respectively, the *a priori* lower and upper bounds on values for $x_i$. The output space $\mathcal{Y}$ is the set of classification labels that can be assigned to any input vector $\mathbf{x} \in \mathcal{X}$. We focus here on binary classification problems. Thus $\mathcal{Y} = \{0, 1\}$. Each example $\mathbf{z} = (\mathbf{x}, y)$ is an input vector $\mathbf{x}$ with its classification label $y \in \mathcal{Y}$. In the probably approximately correct (PAC) setting, we assume that each example $\mathbf{z}$ is generated independently according to the same (but unknown) distribution $D$. The (true) *risk* $R(f)$ of a classifier $f : \mathcal{X} \to \mathcal{Y}$ is defined to be the probability that $f$ misclassifies $\mathbf{z}$ on a random draw according to $D$:

$$R(f) \stackrel{\text{def}}{=} \Pr_{(\mathbf{x}, y) \sim D} (f(\mathbf{x}) \neq y) = \mathbf{E}_{(\mathbf{x}, y) \sim D} I(f(\mathbf{x}) \neq y)$$

where $I(a) = 1$ if predicate $a$ is true and 0 otherwise. Given a training set $S = (\mathbf{z}_1, \ldots, \mathbf{z}_m)$ of $m$ examples, the task of a learning algorithm is to construct a classifier with the smallest possible risk without any information about $D$. To achieve this goal, the learner can compute the *empirical risk* $R_S(f)$ of any given classifier $f$ according to:

$$R_S(f) \stackrel{\text{def}}{=} \frac{1}{m} \sum_{i=1}^{m} I(f(\mathbf{x}_i) \neq y_i) \stackrel{\text{def}}{=} \mathbf{E}_{(\mathbf{x}, y) \sim S} I(f(\mathbf{x}) \neq y)$$

We focus on learning algorithms that construct a *conjunction of rays* from a training set. Each *ray* is just a threshold classifier defined on a single attribute (component) $x_i$. More formally, a ray is identified by an *attribute index* $i \in \{1, \ldots, n\}$, a *threshold value* $t \in [A_i, B_i]$, and a *direction* $d \in \{-1, +1\}$ (that specifies whether class 1 is on the largest or smallest values of $x_i$). Given any input example $\mathbf{x}$, the output $r_{td}^i(\mathbf{x})$ of a ray is defined as:

$$r_{td}^i(\mathbf{x}) \stackrel{\text{def}}{=} \begin{cases} 1 & \text{if} \quad (x_i - t)d > 0 \\ 0 & \text{if} \quad (x_i - t)d \leq 0 \end{cases}$$

To specify a *conjunction of rays* we need first to list all the attributes who's ray is present in the conjunction. For this purpose, we use a vector $\mathbf{i} \stackrel{\text{def}}{=} (i_1, \ldots, i_{|\mathbf{i}|})$

of attribute indices $i_j \in \{1, \ldots, n\}$ such that $i_1 < i_2 < \ldots < i_{|\mathbf{i}|}$ where $|\mathbf{i}|$ is the number of indices present in $\mathbf{i}$ (and thus the number of rays in the conjunction) [1].

To complete the specification of a conjunction of rays, we need a vector $\mathbf{t} = (t_{i_1}, t_{i_2}, \ldots, t_{i_{|\mathbf{i}|}})$ of threshold values and a vector of $\mathbf{d} = (d_{i_1}, d_{i_2}, \ldots, d_{i_{|\mathbf{i}|}})$ of directions where $i_j \in \{1, \ldots, n\}$ for $j \in \{1, \ldots, |\mathbf{i}|\}$. On any input example $\mathbf{x}$, the output $C_{\mathbf{td}}^{\mathbf{i}}(\mathbf{x})$ of a conjunction of rays is given by:

$$C_{\mathbf{td}}^{\mathbf{i}}(\mathbf{x}) \stackrel{\text{def}}{=} \begin{cases} 1 & \text{if} \quad r_{t_j d_j}^j(\mathbf{x}) = 1 \quad \forall j \in \mathbf{i} \\ 0 & \text{if} \quad \exists j \in \mathbf{i} : r_{t_j d_j}^j(\mathbf{x}) = 0 \end{cases}$$

Finally, any algorithm that builds a conjunction can be used to build a disjunction just by exchanging the role of the positive and negative labelled examples. Due to lack of space, we describe here only the case of a conjunction.

## 3  A PAC-Bayes Risk Bound

The PAC-Bayes approach, initiated by McAllester (1999), aims at providing PAC guarantees to "Bayesian" learning algorithms. These algorithms are specified in terms of a *prior distribution* $P$ over a space of classifiers that characterizes our prior belief about good classifiers (before the observation of the data) and a *posterior distribution* $Q$ (over the same space of classifiers) that takes into account the additional information provided by the training data. A remarkable result that came out from this line of research, known as the "PAC-Bayes theorem", provides a tight upper bound on the risk of a stochastic classifier called the *Gibbs classifier*. Given an input example $\mathbf{x}$, the label $G_Q(\mathbf{x})$ assigned to $\mathbf{x}$ by the Gibbs classifier is defined by the following process. We first choose a classifier $h$ according to the posterior distribution $Q$ and then use $h$ to assign the label $h(\mathbf{x})$ to $\mathbf{x}$. The risk of $G_Q$ is defined as the expected risk of classifiers drawn according to $Q$:

$$R(G_Q) \stackrel{\text{def}}{=} \mathbf{E}_{h \sim Q} R(h) = \mathbf{E}_{h \sim Q} \mathbf{E}_{(\mathbf{x},y) \sim D} I(f(\mathbf{x}) \neq y)$$

The PAC-Bayes theorem was first proposed by McAllester (2003). The version presented here is due to Seeger (2002) and Langford (2003).

**Theorem 1** *Given any space $\mathcal{H}$ of classifiers. For any data-independent prior distribution $P$ over $\mathcal{H}$ and for any (possibly data-dependent) posterior distribution $Q$ over $\mathcal{H}$, with probability at least $1 - \delta$ over the random draws of training sets $S$ of $m$ examples:*

$$\mathrm{kl}(R_S(G_Q) \| R(G_Q)) \leq \frac{\mathrm{KL}(Q \| P) + \ln \frac{m+1}{\delta}}{m}$$

*where $\mathrm{KL}(Q \| P)$ is the Kullback-Leibler divergence between distributions[2] $Q$ and $P$:*

$$\mathrm{KL}(Q \| P) \stackrel{\text{def}}{=} \mathbf{E}_{h \sim Q} \ln \frac{Q(h)}{P(h)}$$

*and where $\mathrm{kl}(q \| p)$ is the Kullback-Leibler divergence between the Bernoulli distributions with probabilities of success $q$ and $p$:*

$$\mathrm{kl}(q \| p) \stackrel{\text{def}}{=} q \ln \frac{q}{p} + (1 - q) \ln \frac{1-q}{1-p} \quad \text{for } q < p$$

The bound given by the PAC-Bayes theorem for the risk of Gibbs classifiers can be turned into a bound for the risk of Bayes classifiers in the following way. Given a posterior distribution $Q$, the Bayes classifier $B_Q$ performs a majority vote (under measure $Q$) of binary classifiers in $\mathcal{H}$. When $B_Q$ misclassifies an example $\mathbf{x}$, at least half of the binary classifiers (under measure $Q$) misclassifies $\mathbf{x}$. It follows that the error rate of $G_Q$ is at least half of the error rate of $B_Q$. Hence $R(B_Q) \leq 2R(G_Q)$.

In our case, we have seen that ray conjunctions are specified in terms of a mixture of discrete parameters $\mathbf{i}$ and $\mathbf{d}$ and continuous parameters $\mathbf{t}$. If we denote by $P_{\mathbf{i},\mathbf{d}}(\mathbf{t})$ the probability density function associated with a prior $P$ over the class of ray conjunctions, we consider here priors of the form:

$$P_{\mathbf{i},\mathbf{d}}(\mathbf{t}) = \frac{1}{\binom{n}{|\mathbf{i}|}} p(|\mathbf{i}|) \frac{1}{2^{|\mathbf{i}|}} \prod_{j \in \mathbf{i}} \frac{1}{B_j - A_j} \quad ; \quad \forall t_j \in [A_j, B_j]$$

If $\mathcal{I}$ denotes the set of all $2^n$ possible attribute index vectors and $\mathcal{D}_{\mathbf{i}}$ denotes de set of all $2^{|\mathbf{i}|}$ binary direction vectors $\mathbf{d}$ of dimension $|\mathbf{i}|$, we have that:

$$\sum_{\mathbf{i} \in \mathcal{I}} \sum_{\mathbf{d} \in \mathcal{D}_{\mathbf{i}}} \prod_{j \in \mathbf{i}} \int_{A_j}^{B_j} dt_j P_{\mathbf{i},\mathbf{d}}(\mathbf{t}) = 1$$

whenever $\sum_{e=0}^{n} p(e) = 1$.

The reasons motivating this choice for the prior are the following. The first two factors come from the belief that the final classifier, constructed from the group of attributes specified by $\mathbf{i}$, should depend only on the number $|\mathbf{i}|$ of attributes in this group. If we have complete ignorance about the number of rays the final classifier is likely to have, we should choose $p(e) = 1/(n+1)$ for $e \in \{0, 1, \ldots, n\}$. However, we should choose a $p$ that decreases as we increase $e$ if we have reasons to believe that the number of rays of the final classifier will be much smaller than $n$. The third factor of $P_{\mathbf{i},\mathbf{d}}(\mathbf{t})$ gives equal prior probabilities for each of the two possible values of direction $d_j$. Finally, for each ray, every possible threshold value $t$ should have the same prior probability of being chosen if we do not have any prior knowledge that would favor some values over the others. Since each attribute value $x_i$ is constrained, a priori, to be in $[A_i, B_i]$, we have chosen a uniform probability density on $[A_i, B_i]$ for each $t_i$ such that $i \in \mathbf{i}$. This explains the last factors of $P_{\mathbf{i},\mathbf{d}}(\mathbf{t})$.

Given a training set $S$, the learner will choose an attribute group $\mathbf{i}$ and a direction vector $\mathbf{d}$. For each attribute $x_i \in [A_i, B_i] : i \in \mathbf{i}$, a margin interval $[a_i, b_i] \subseteq [A_i, B_i]$ will also be chosen by the learner. A deterministic ray-conjunction classifier is then specified by choosing the thresholds values $t_i \in [a_i, b_i]$. It is tempting at this point to choose $t_i = (a_i + b_i)/2 \ \forall i \in \mathbf{i}$ (i.e., in the middle of each interval). However, we will see shortly that the PAC-Bayes theorem offers a better guarantee for another type of deterministic classifier.

The Gibbs classifier is defined with a posterior distribution $Q$ having all its weight on the same $\mathbf{i}$ and $\mathbf{d}$ as chosen by the learner but where each $t_i$ is uniformly chosen in $[a_i, b_i]$. The KL divergence between this posterior $Q$ and the prior $P$ is then given by:

$$
\begin{aligned}
KL(Q\|P) &= \prod_{j \in \mathbf{i}} \int_{a_j}^{b_j} \frac{dt_j}{b_j - a_j} \ln\left( \frac{\prod_{i \in \mathbf{i}} (b_i - a_i)^{-1}}{P_{\mathbf{i},\mathbf{d}}(\mathbf{t})} \right) \\
&= \ln\binom{n}{|\mathbf{i}|} + \ln\left( \frac{1}{p(|\mathbf{i}|)} \right) + |\mathbf{i}| \ln(2) + \sum_{i \in \mathbf{i}} \ln\left( \frac{B_i - A_i}{b_i - a_i} \right)
\end{aligned}
$$

Hence, we see that the KL divergence between the "continuous components" of $Q$ and $P$ (given by the last term) vanishes when $[a_i, b_i] = [A_i, B_i] \ \forall i \in \mathbf{i}$. Furthermore,

the KL divergence between the "discrete components" of $Q$ and $P$ is small for small values of $|\mathbf{i}|$ (whenever $p(|\mathbf{i}|)$ is not too small). *Hence, this KL divergence between our choices for $Q$ and $P$ exhibits a tradeoff between margins (large values of $b_i - a_i$) and sparsity (small value of $|\mathbf{i}|$) for Gibbs classifiers.* According to Theorem 1, the Gibbs classifier with the smallest guarantee of risk $R(G_Q)$ should minimize a non trivial combination of $KL(Q\|P)$ (margins-sparsity tradeoff) and empirical risk $R_S(G_Q)$.

Since the posterior $Q$ is identified by an attribute group vector $\mathbf{i}$, a direction vector $\mathbf{d}$, and intervals $[a_i, b_i]\ \forall i \in \mathbf{i}$, we will refer to the Gibbs classifier $G_Q$ by $G_{\mathbf{ab}}^{\mathbf{id}}$ where $\mathbf{a}$ and $\mathbf{b}$ are the vectors formed by the unions of $a_i$s and $b_i$s respectively. We can obtain a closed-form expression for $R_S(G_{\mathbf{ab}}^{\mathbf{id}})$ by first considering the risk $R_{(\mathbf{x},y)}(G_{\mathbf{ab}}^{\mathbf{id}})$ on a single example $(\mathbf{x}, y)$ since $R_S(G_{\mathbf{ab}}^{\mathbf{id}}) = \mathbf{E}_{(\mathbf{x},y)\sim S} R_{(\mathbf{x},y)}(G_{\mathbf{ab}}^{\mathbf{id}})$. From our definition for $Q$, we find that:

$$R_{(\mathbf{x},y)}(G_{\mathbf{ab}}^{\mathbf{id}}) = (1 - 2y)\left[\prod_{i\in\mathbf{i}} \sigma_{a_i,b_i}^{d_i}(x_i) - y\right] \tag{1}$$

where we have used the following piece-wise linear functions:

$$\sigma_{a,b}^+(x) \stackrel{\text{def}}{=} \begin{cases} 0 & \text{if}\quad x < a \\ \frac{x-a}{b-a} & \text{if}\quad a \leq x \leq b \\ 1 & \text{if}\quad b < x \end{cases} \quad;\quad \sigma_{a,b}^-(x) \stackrel{\text{def}}{=} \begin{cases} 1 & \text{if}\quad x < a \\ \frac{b-x}{b-a} & \text{if}\quad a \leq x \leq b \\ 0 & \text{if}\quad b < x \end{cases} \tag{2}$$

Hence we notice that $R_{(\mathbf{x},1)}(G_{\mathbf{ab}}^{\mathbf{id}}) = 1$ (and $R_{(\mathbf{x},0)}(G_{\mathbf{ab}}^{\mathbf{id}}) = 0$) whenever there exist $i \in \mathbf{i} : \sigma_{a_i,b_i}^{d_i}(x_i) = 0$. This occurs iff there exists a ray which outputs 0 on $\mathbf{x}$. We can also verify that the expression for $R_{(\mathbf{x},y)}(C_{\mathbf{td}}^{\mathbf{i}})$ is identical to the expression for $R_{(\mathbf{x},y)}(G_{\mathbf{ab}}^{\mathbf{id}})$ except that the piece-wise linear functions $\sigma_{a_i,b_i}^{d_i}(x_i)$ are replaced by the indicator functions $I((x_i - t_i)d_i > 0)$.

The PAC-Bayes theorem provides a risk bound for the Gibbs classifier $G_{\mathbf{ab}}^{\mathbf{id}}$. Since the Bayes classifier $B_{\mathbf{ab}}^{\mathbf{id}}$ just performs a majority vote under the same posterior distribution as the one used by $G_{\mathbf{ab}}^{\mathbf{id}}$, we have that $B_{\mathbf{ab}}^{\mathbf{id}}(\mathbf{x}) = 1$ iff the probability that $G_{\mathbf{ab}}^{\mathbf{id}}$ classifies $\mathbf{x}$ as positive exceeds $1/2$. Hence, it follows that

$$B_{\mathbf{ab}}^{\mathbf{id}}(\mathbf{x}) = \begin{cases} 1 & \text{if}\quad \prod_{i\in\mathbf{i}} \sigma_{a_i,b_i}^{d_i}(x_i) > 1/2 \\ 0 & \text{if}\quad \prod_{i\in\mathbf{i}} \sigma_{a_i,b_i}^{d_i}(x_i) \leq 1/2 \end{cases} \tag{3}$$

Note that $B_{\mathbf{ab}}^{\mathbf{id}}$ has an *hyperbolic* decision surface. Consequently, $B_{\mathbf{ab}}^{\mathbf{id}}$ is not representable as a conjunction of rays. There is, however, no computational difficulty at obtaining the output of $B_{\mathbf{ab}}^{\mathbf{id}}(\mathbf{x})$ for any $\mathbf{x} \in \mathcal{X}$.

From the relation between $B_{\mathbf{ab}}^{\mathbf{id}}$ and $G_{\mathbf{ab}}^{\mathbf{id}}$, it also follows that $R_{(\mathbf{x},y)}(B_{\mathbf{ab}}^{\mathbf{id}}) \leq 2R_{(\mathbf{x},y)}(G_{\mathbf{ab}}^{\mathbf{id}})$ for any $(\mathbf{x}, y)$. Consequently, $R(B_{\mathbf{ab}}^{\mathbf{id}}) \leq 2R(G_{\mathbf{ab}}^{\mathbf{id}})$. Hence, we have our main theorem:

**Theorem 2** *Given all our previous definitions, for any $\delta \in (0, 1]$, and for any $p$ satisfying $\sum_{e=0}^n p(e) = 1$, we have:*

$$\Pr_{S\sim D^m}\left(\forall \mathbf{i}, \mathbf{d}, \mathbf{a}, \mathbf{b}\colon R(G_{\mathbf{ab}}^{\mathbf{id}}) \leq \sup\left\{\epsilon\colon \mathrm{kl}(R_S(G_{\mathbf{ab}}^{\mathbf{id}})\|\epsilon) \leq \frac{1}{m}\left[\ln\binom{n}{|\mathbf{i}|} + \right.\right.\right.$$

$$\left.\left.\left. + |\mathbf{i}|\ln(2) + \ln\left(\frac{1}{p(|\mathbf{i}|)}\right) + \sum_{i\in\mathbf{i}}\ln\left(\frac{B_i - A_i}{b_i - a_i}\right) + \ln\frac{m+1}{\delta}\right]\right\}\right) \geq 1 - \delta$$

*Furthermore: $R(B_{\mathbf{ab}}^{\mathbf{id}}) \leq 2R(G_{\mathbf{ab}}^{\mathbf{id}}) \quad \forall \mathbf{i}, \mathbf{d}, \mathbf{a}, \mathbf{b}$.*

# 4    A Soft Greedy Learning Algorithm

Theorem 2 suggests that the learner should try to find the Bayes classifier $B_{\mathbf{ab}}^{\mathbf{id}}$ that uses a small number of attributes (*i.e.*, a small $|\mathbf{i}|$), each with a large separating margin $(b_i - a_i)$, while keeping the empirical Gibbs risk $R_S(G_{\mathbf{ab}}^{\mathbf{id}})$ at a low value. To achieve this goal, we have adapted the greedy algorithm for the set covering machine (SCM) proposed by Marchand and Shawe-Taylor (2002). It consists of choosing the feature (here a ray) $i$ with the largest *utility* $U_i$ where:

$$U_i = |Q_i| - p|R_i|$$

where $Q_i$ is the set of negative examples covered (classified as 0) by feature $i$, $R_i$ is the set of positive examples misclassified by this feature, and $p$ is a learning parameter that gives a penalty $p$ for each misclassified positive example. Once the feature with the largest $U_i$ is found, we remove $Q_i$ and $P_i$ from the training set $S$ and then repeat (on the remaining examples) until either no more negative examples are present or that a maximum number $s$ of features has been reached.

In our case, however, we need to keep the Gibbs risk on $S$ low instead of the risk of a deterministic classifier. Since the Gibbs risk is a "soft measure" that uses the piece-wise linear functions $\sigma_{a,b}^d$ instead of the "hard" indicator functions, we need a "softer" version of the utility function $U_i$. Indeed, a negative example that falls in the linear region of a $\sigma_{a,b}^d$ is in fact partly covered. Following this observation, let $\mathbf{k}$ be the vector of indices of the attributes that we have used so far for the construction of the classifier. Let us first define the *covering value* $\mathcal{C}(G_{\mathbf{ab}}^{\mathbf{kd}})$ of $G_{\mathbf{ab}}^{\mathbf{kd}}$ by the "amount" of negative examples assigned to class 0 by $G_{\mathbf{ab}}^{\mathbf{kd}}$:

$$\mathcal{C}(G_{\mathbf{ab}}^{\mathbf{kd}}) \quad \stackrel{\text{def}}{=} \quad \sum_{(\mathbf{x},y)\in S} (1-y) \left[ 1 - \prod_{j\in\mathbf{k}} \sigma_{a_j,b_j}^{d_j}(x_j) \right]$$

We also define the *positive-side error* $\mathcal{E}(G_{\mathbf{ab}}^{\mathbf{kd}})$ of $G_{\mathbf{ab}}^{\mathbf{kd}}$ as the "amount" of positive examples assigned to class 0 :

$$\mathcal{E}(G_{\mathbf{ab}}^{\mathbf{kd}}) \quad \stackrel{\text{def}}{=} \quad \sum_{(\mathbf{x},y)\in S} y \left[ 1 - \prod_{j\in\mathbf{k}} \sigma_{a_j,b_j}^{d_j}(x_j) \right]$$

We now want to add another ray on another attribute, call it $i$, to obtain a new vector $\mathbf{k}'$ containing this new attribute in addition to those present in $\mathbf{k}$. Hence, we now introduce the *covering contribution* of ray $i$ as:

$$\mathcal{C}_{\mathbf{ab}}^{\mathbf{kd}}(i) \quad \stackrel{\text{def}}{=} \quad \mathcal{C}(G_{\mathbf{a'b'}}^{\mathbf{k'd'}}) - \mathcal{C}(G_{\mathbf{ab}}^{\mathbf{kd}}) \; = \; \sum_{(\mathbf{x},y)\in S} (1-y) \left[ 1 - \sigma_{a_i,b_i}^{d_i}(x_i) \right] \prod_{j\in\mathbf{k}} \sigma_{a_j,b_j}^{d_j}(x_j)$$

and the *positive-side error contribution* of ray $i$ as:

$$\mathcal{E}_{\mathbf{ab}}^{\mathbf{kd}}(i) \quad \stackrel{\text{def}}{=} \quad \mathcal{E}(G_{\mathbf{a'b'}}^{\mathbf{k'd'}}) - \mathcal{E}(G_{\mathbf{ab}}^{\mathbf{kd}}) \; = \; \sum_{(\mathbf{x},y)\in S} y \left[ 1 - \sigma_{a_i,b_i}^{d_i}(x_i) \right] \prod_{j\in\mathbf{k}} \sigma_{a_j,b_j}^{d_j}(x_j)$$

Typically, the covering contribution of ray $i$ should increase its "utility" and its positive-side error should decrease it. Moreover, we want to decrease the "utility" of ray $i$ by an amount which would become large whenever it has a small separating margin. Our expression for $KL(Q\|P)$ suggests that this amount should be proportional to $\ln((B_i - A_i)/(b_i - a_i))$. Furthermore we should compare this margin term with the *fraction* of the remaining negative examples that ray $i$ has covered

(instead of the absolute amount of negative examples covered). Hence the covering contribution $\mathcal{C}_{\mathbf{ab}}^{\mathbf{kd}}(i)$ of ray $i$ should be divided by the amount $\mathcal{N}_{\mathbf{ab}}^{\mathbf{kd}}$ of negative examples that *remains* to be covered before considering ray $i$:

$$\mathcal{N}_{\mathbf{ab}}^{\mathbf{kd}} \stackrel{\text{def}}{=} \sum_{(\mathbf{x},y) \in S} (1-y) \prod_{j \in \mathbf{k}} \sigma_{a_j,b_j}^{d_j}(x_j)$$

which is simply the amount of negative examples that have been assigned to class 1 by $G_{\mathbf{ab}}^{\mathbf{kd}}$. If $P$ denotes the set of positive examples, we define the *utility $U_{\mathbf{ab}}^{\mathbf{kd}}(i)$ of adding ray $i$ to $G_{\mathbf{ab}}^{\mathbf{kd}}$* as:

$$U_{\mathbf{ab}}^{\mathbf{kd}}(i) \stackrel{\text{def}}{=} \frac{\mathcal{C}_{\mathbf{ab}}^{\mathbf{kd}}(i)}{\mathcal{N}_{\mathbf{ab}}^{\mathbf{kd}}} - p\frac{\mathcal{E}_{\mathbf{ab}}^{\mathbf{kd}}(i)}{|P|} - \eta \ln \frac{B_i - A_i}{b_i - a_i}$$

where parameter $p$ represents the *penalty* of misclassifying a positive example and $\eta$ is another parameter that controls the importance of having a large margin. These learning parameters can be chosen by cross-validation. For fixed values of these parameters, the "soft greedy" algorithm simply consists of adding, to the current Gibbs classifier, a ray with maximum added utility until either the maximum number $s$ of rays has been reached or that all the negative examples have been (totally) covered. It is understood that, during this soft greedy algorithm, we can remove an example $(\mathbf{x}, y)$ from $S$ whenever it is totally covered. This occurs whenever $\prod_{j \in \mathbf{k}} \sigma_{a_j,b_j}^{d_j}(x_j) = 0$.

# 5  Results for Classification of DNA Micro-Arrays

We have tested the soft greedy learning algorithm on the four DNA micro-array data sets shown in Table 1. The *colon tumor* data set (Alon et al., 1999) provides the expression levels of 40 tumor and 22 normal colon tissues measured for 6500 human genes. The *ALL/AML* data set (Golub et al., 1999) provides the expression levels of 7129 human genes for 47 samples of patients with acute lymphoblastic leukemia (ALL) and 25 samples of patients with acute myeloid leukemia (AML). The *B_MD* and *C_MD* data sets (Pomeroy et al., 2002) are micro-array samples containing the expression levels of 6817 human genes. Data set B contains 25 classic and 9 desmoplastic medulloblastomas whereas data set C contains 39 medulloblastomas survivors and 21 treatment failures (non-survivors).

We have compared the soft greedy learning algorithm with a linear-kernel soft-margin SVM trained both on all the attributes (gene expressions) and on a subset of attributes chosen by the filter method of Golub et al. (1999). It consists of ranking the attributes as function of the difference between the positive-example mean and the negative-example mean and then use only the first $\ell$ attributes. The resulting learning algorithm, named SVM+gs in Table 1, is basically the one used by Furey et al. (2000) for the same task. Guyon et al. (2002) claimed obtaining better results with the recursive feature elimination method but, as pointed out by Ambroise and McLachlan (2002), their work contained a methodological flaw and, consequently, the superiority of this wrapper method is questionable.

Each algorithm was tested with the 5-fold cross validation (CV) method. Each of the five training sets and testing sets was the same for all algorithms. The learning parameters of all algorithms and the gene subsets (for SVM+gs) were chosen from the training sets *only*. This was done by performing a second (nested) 5-fold CV on each training set. For the gene subset selection procedure of SVM+gs, we have considered the first $\ell = 2^i$ genes (for $i = 0, 1, \ldots, 12$) ranked according to the criterion of Golub et al. (1999) and have chosen the $i$ value that gave the smallest 5-fold CV error on the training set.

| Data Set | | SVM | SVM+gs | | Soft Greedy | | | | |
|---|---|---|---|---|---|---|---|---|---|
| Name | #exs | errs | errs | size | ratio | size | G-errs | B-errs | Bound |
| Colon | 62 | 12 | 11 | 256 | 0.42 | 1 | 12 | 9 | 18 |
| B_MD | 34 | 12 | 6 | 32 | 0.10 | 1 | 6 | 6 | 20 |
| C_MD | 60 | 29 | 21 | 1024 | 0.077 | 3 | 24 | 22 | 40 |
| ALL/AML | 72 | 18 | 10 | 64 | 0.002 | 2 | 19 | 17 | 38 |

Table 1: DNA micro-array data sets and results.

For each algorithm, the "errs" columns of Table 1 contain the 5-fold CV error expressed as the sum of errors over the five testing sets and the "size" columns contain the number of attributes used by the classifier averaged over the five testing sets. The "G-err" and "B-err" columns refer to the Gibbs and Bayes error rates. The "ratio" column refers to the average value of $(b_i - a_i)/(B_i - A_i)$ obtained for the rays used by classifiers and the "bound" column refers to the average risk bound of Theorem 2 multiplied by the total number of examples. We see that the gene selection filter generally improves the error rate of SVM and that the Bayes error rate is slightly better than the Gibbs error rate. Finally, the error rates of Bayes and SVM+gs are competitive but the number of genes selected by the soft greedy algorithm is always *much* smaller.

## Footnotes

[1] Although it is possible to use up to two rays on any attribute, we limit ourselves here to the case where each attribute can be used for only one ray.

[2] Here $Q(h)$ denotes the probability density function associated to $Q$, evaluated at $h$.

# References

U. Alon, N. Barkai, D.A. Notterman, K. Gish, S. Ybarra, D. Mack, and A.J. Levine. Broad patterns of gene expression revealed by clustering analysis of tumor and normal colon tissues probed by oligonucleotide arrays. *PNAS USA*, 96:6745–6750, 1999.

C. Ambroise and G. J. McLachlan. Selection bias in gene extraction on the basis of microarray gene-expression data. *Proc. Natl. Acad. Sci. USA*, 99:6562–6566, 2002.

T. S. Furey, N. Cristianini, N. Duffy, D. W. Bednarski, M. Schummer, and D. Haussler. Support vector machine classification and validation of cancer tissue samples using microarray expression data. *Bioinformatics*, 16:906–914, 2000.

T.R. Golub, D.K. Slonim, and Many More Authors. Molecular classification of cancer: class discovery and class prediction by gene expression monitoring. *Science*, 286:531–537, 1999.

I. Guyon, J. Weston, S. Barnhill, and V. Vapnik. Gene selection for cancer classification using support vector machines. *Machine Learning*, 46:389–422, 2002.

D. Haussler. Quantifying inductive bias: AI learning algorithms and Valiant's learning framework. *Artificial Intelligence*, 36:177–221, 1988.

John Langford. Tutorial on practical prediction theory for classification. `http://hunch.net/~jl/projects/prediction_bounds/tutorial/tutorial.ps`, 2003.

N. Littlestone. Learning quickly when irrelevant attributes abound: A new linear-threshold algorithm. *Machine Learning*, 2(4):285–318, 1988.

Mario Marchand and John Shawe-Taylor. The set covering machine. *Journal of Machine Learning Reasearch*, 3:723–746, 2002.

David McAllester. Some PAC-Bayesian theorems. *Machine Learning*, 37:355–363, 1999.

David McAllester. PAC-Bayesian stochastic model selection. *Machine Learning*, 51:5–21, 2003. A priliminary version appeared in proceedings of COLT'99.

S. L. Pomeroy, P. Tamayo, and Many More Authors. Prediction of central nervous system embryonal tumour outcome based on gene expression. *Nature*, 415:436–442, 2002.

Matthias Seeger. PAC-Bayesian generalization bounds for gaussian processes. *Journal of Machine Learning Research*, 3:233–269, 2002.
